# Environmental statistics and the trade-off between model-based and TD learning in humans

**Dylan A. Simon**
Department of Psychology
New York University
New York, NY 10003
dylex@nyu.edu

**Nathaniel D. Daw**
Center for Neural Science and Department of Psychology
New York University
New York, NY 10003
nathaniel.daw@nyu.edu

## Abstract

There is much evidence that humans and other animals utilize a combination of model-based and model-free RL methods. Although it has been proposed that these systems may dominate according to their relative statistical efficiency in different circumstances, there is little specific evidence — especially in humans — as to the details of this trade-off. Accordingly, we examine the relative performance of different RL approaches under situations in which the statistics of reward are differentially noisy and volatile. Using theory and simulation, we show that model-free TD learning is relatively most disadvantaged in cases of high volatility and low noise. We present data from a decision-making experiment manipulating these parameters, showing that humans shift learning strategies in accord with these predictions. The statistical circumstances favoring model-based RL are also those that promote a high learning rate, which helps explain why, in psychology, the distinction between these strategies is traditionally conceived in terms of rule-based vs. incremental learning.

## 1   Introduction

There are many suggestions that humans and other animals employ multiple approaches to learned decision making [1]. Precisely delineating these approaches is key to understanding human decision systems, especially since many problems of behavioral control such as addiction have been attributed to partial failures of one component [2]. In particular, understanding the trade-offs between approaches in order to bring them under experimental control is critical for isolating their unique contributions and ultimately correcting maladaptive behavior. Psychologists primarily distinguish between declarative rule learning and more incremental learning of stimulus-response (S–R) habits across a broad range of tasks [3, 4]. They have shown that large problem spaces, probabilistic feedback (as in the weather prediction task), and difficult to verbalize rules (as in information integration tasks from category learning) all seem to promote the use of a habit learning system [5, 6, 7, 8, 9]. The alternative strategies, which these same manipulations disfavor, are often described as imputing (inherently deterministic) 'rules' or 'maps', and are potentially supported by dissociable neural systems also involved in memory for one-shot episodes [10].

Neuroscientists studying rats have focused on more specific tasks that test whether animals are sensitive to changes in the outcome contingency or value of actions. For instance, under different task circumstances or following different brain lesions, rats are more or less willing to continue working for a devalued food reward [11]. In terms of reinforcement learning (RL) theories, such evidence has been proposed to reflect a distinction between parallel systems for model-based vs. model-free RL [12, 13]: a world model permits updating a policy following a change in food value, while model-free methods preclude this.

Intuitively, S–R habits correspond well to the policies learned by TD methods such as actor/critic [14, 15], and rule-based cognitive planning strategies seem to mirror model-based algorithms. However, the implication that this distinction fundamentally concerns the use or non-use of a world model in representation and algorithm seems somewhat at odds with the conception in psychology. Specifically, neither the gradation of update (i.e., incremental vs. abrupt) nor the nature of representation (i.e., verbalizable rules) posited in the declarative system seem obviously related to the model-use distinction. Although there have been some suggestions about how episodic memory may support TD learning [16], a world model as conceived in RL is typically inherently probabilistic, so as to support computing expected action values in stochastic environments, and thus must be learned by incrementally composing multiple experiences. It has also been suggested that episodic memory supports yet a third decision strategy distinct from both model-based and model-free [17], although there is no experimental evidence for such a triple dissociation or in particular for a separation between the putative episodic and model-based controllers.

Here we suggest that an explanation for this mismatch may follow from the circumstances under which each RL approach dominates. It has previously been proposed that model-free and model-based reasoning should be traded off according to their relative statistical efficiency (proxied by uncertainty) in different circumstances [13]. In fact, what ultimately matters to a decision-maker is relative advantage in terms of reward [18]. Focusing specifically on task statistics, we extend the uncertainty framework to investigate under what circumstances the performance of a model-based system excels sufficiently to make it worthwhile.

When the environment is completely static, TD is well known to converge to the optimal policy almost as quickly as model-based approaches [19], and so environmental change must be key to understanding its computational disadvantages. Primarily, model-free Monte Carlo (MC) methods such as TD are unable to propagate learned information around the state space efficiently, and in particular to generalize to states not observed in the current trajectory. This is not the only way in which MC methods learn slowly, however: they must also take samples of outcomes and average over them. This process introduces additional noise to the sampling process which must be averaged over, as observational deviations resulting from the learner's own choice variability or transition stochasticity in the environment are confounded with variability in immediate rewards. In effect, this averaging imposes an upper bound on the learning rate needed to achieve reasonable performance, and, correspondingly, on how well it can keep up with task volatility.

Conversely, the key benefit of model-based reasoning lies in its ability to react quickly to change, applying single-trial experience flexibly in order to construct values. We provide a more formal argument of this observation in MDPs with dynamic rewards and static transitions, and find that the environments in which TD is most impaired are those with frequent changes and little noise. This suggests a strategy by which these two approaches should optimally trade-off, which we test empirically using a decision task in humans while manipulating reward statistics. The high-volatility environments in which model-based learning dominates are also those in which a learning rate near one optimally applies. This may explain why a model-based system is associated with or perhaps specialized for rapid, declarative rule learning.

## 2 Theory

Model-free and model-based methods differ in their strategies for estimating action values from samples. One key disadvantage of Monte Carlo sampling of long-run values in an MDP, relative to model-based RL (in which immediate rewards are sampled and aggregated according to the sampled transition dynamics), is the need to average samples over both reward and state transition stochasticity. This impairs its ability to track changes in the underlying MDP, with the disadvantage most pronounced in situations of high volatility and low noise.

Below, we develop the intuition for this disadvantage by applying Kalman filter analysis [20] to examine uncertainties in the simplest possible MDP that exhibits the issue. Specifically, consider a state with two actions, each associated with a pair of terminal states. Each action leads to one of the two states with equal probability, and each of the four terminal states is associated with a reward. The rewards are stochastic and diffusing, according to a Gaussian process, and the transitions are fixed. We consider the uncertainty and reward achievable as a function of the volatility and observation noise. We have here made some simplifications in order to make the intuition as clear as possible:

that each trajectory has only a single state transition and reward; that in the steady state the static transition matrix has been fully learned; and that all analyzed distributions are Gaussian. We test some of these assumptions empirically in section 3 by showing that the same pattern holds in more complex tasks.

## 2.1 Model

In general $X_t(i)$ or just $X$ will refer to an actual sample of the $i$th variable (e.g., reward or value) at time $t$, $\bar{X}$ refers to the (latent) true mean of $X$, and $\hat{X}$ refers to estimates of $\bar{X}$ made by the learning process. Given i.i.d. Gaussian diffusion processes on each value, $X_t(i)$, described by:

$$\sigma^2 = \left\langle (\bar{X}_{t+1}(i) - \bar{X}_t(i))^2 \right\rangle \qquad \text{diffusion or volatility,} \qquad (1)$$

$$\varepsilon^2 = \left\langle (X_t(i) - \bar{X}_t(i))^2 \right\rangle \qquad \text{and observation noise,} \qquad (2)$$

the optimal learning rate that achieves the minimal uncertainty (from the Kalman gain) is:

$$\alpha^* = \frac{\sigma\sqrt{\sigma^2 + 4\varepsilon^2} - \sigma^2}{2\varepsilon^2} \qquad (3)$$

Note that this function is monotonically increasing with $\sigma$ and decreasing with $\varepsilon$ (and in particular, $\alpha^* \to 1$ as $\varepsilon \to 0$). When using this learning rate the resulting asymptotic uncertainty (variance of estimates) will be:

$$U_X(\alpha^*) = \left\langle (\hat{X} - \bar{X})^2 \right\rangle = \frac{\sigma\sqrt{\sigma^2 + 4\varepsilon^2} + \sigma^2}{2} \qquad (4)$$

This, as expected, increases monotonically in both parameters.

What often matters, however, is identifying the highest of multiple values, e.g., $\bar{X}(i)$ and $\bar{X}(j)$. If $\bar{X}(i) - \bar{X}(j) = d$, the marginal value of the choice will be $\pm d$. Given some uncertainty, $U$, the probability of this choice, i.e., $\hat{X}(i) > \hat{X}(j)$, compared to chance is:

$$c(U) = 2 \int_{-\infty}^{\infty} \phi\left(x - \frac{d}{\sqrt{U}}\right) \Phi(x) \mathrm{d}x - 1 \qquad (5)$$

(Where $\phi$ and $\Phi$ are the density and distribution functions for the standard normal.) The resulting value of the choice is thus $c(U)d$. While $c$ is flat at 1 as $U \to 0$, it shrinks as $\Theta(1/\sqrt{U})$ (since $\phi'(0) = 0$). Our goal is now to determine $c(U_Q)$ for each algorithm.

## 2.2 Value estimation

Consider the value of one of the actions in our two-action MDP which leads to state A or B. Here, the true expected value of the choice is $\bar{Q} = \frac{\bar{R}(\mathsf{A}) + \bar{R}(\mathsf{B})}{2}$. If each reward is changing according to the Gaussian diffusion process described above, this will induce a change process on $Q$. A model-based system that has fully learned the transition dynamics will be able to estimate $\hat{R}(\mathsf{A})$ and $\hat{R}(\mathsf{B})$ separately, and thus take the expectation to produce $\hat{Q}$. By assuming each reward is sampled equally often and adopting the appropriate effective $\sigma$, the resulting uncertainty of this expectation, $U_{\mathrm{MB}}$, follows Equation 4, with $X = Q$.

On the other hand, a Monte Carlo system that must take samples over transitions will observe $Q = R(\mathsf{A})$ or $Q = R(\mathsf{B})$. If $\left| \bar{R}(\mathsf{A}) - \bar{R}(\mathsf{B}) \right| = d$, it will observe an additional variance of $\frac{d^2}{4}$ from the mixture of the two reward distributions. Treating this noise as Gaussian and adding it to the noise of the rewards, this decreases the optimal learning rate and increases the minimal uncertainty to:

$$U_{\mathrm{MC}} = \left\langle (\hat{Q} - \bar{Q})^2 \right\rangle = \frac{\sigma\sqrt{\sigma^2 + d^2 + 4\varepsilon^2} + \sigma^2}{2} \qquad (6)$$

Other forms of stochasticity, whether from changing policies or more complex MDPs, will similarly inflate the effective noise term, albeit with a different form.

Clearly $U_{\mathrm{MC}} \geq U_{\mathrm{MB}}$. However, the more relevant measure is how these uncertainties translate into values [18]. For this we want to compare their relative success rates, $c(U)$ from Equation 5, which scale directly to outcome. The relative advantage of the model-based (MB) approach, $c(U_{\mathrm{MB}}) - $

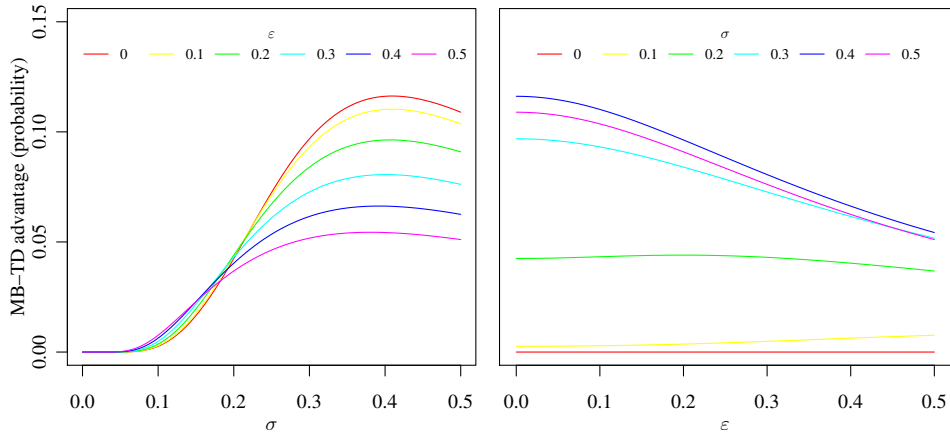

Figure 1: Difference in theoretical success rate between MB and MC

$c(U_{\mathrm{MC}})$, is plotted in Figure 1 for an arbitrary reward deviation $d = 1$. As expected, as either the volatility or noise parameter gets very large and the task gets harder, the uncertainty increases, performance approaches chance, and the relative advantage vanishes. However, for reasonable sizes of $\sigma$, the model-based advantage first increases to a peak as $\sigma$ increases, which is largest for small values of $\varepsilon$. No comparable increasing advantage is seen for model-based valuation for increasing $\varepsilon$.

While these techniques may also be extended more generally to other MDPs (see Supplemental Materials), the core observation presented above should illuminate the remainder of our discussion.

## 3   Simulation

To examine our qualitative predictions in a more realistic setting, we simulated randomly generated MDPs with 8 states, 2 actions, and transition and reward functions following the assumptions given in the previous section, with the addition of a contractive factor on rewards, $\varphi$, to prevent divergence:

$$\bar{R}_0(s,a) \sim \mathcal{N}(0,1) \qquad\qquad \text{stationary distribution}$$
$$\varphi = \sqrt{1-\sigma^2} \qquad\qquad \mathrm{var}\,\bar{R} = 1$$
$$\bar{R}_t(s,a) = \varphi\bar{R}_{t-1}(s,a) + w_t(s,a) \qquad\qquad w_t(s,a) \sim \mathcal{N}(0,\sigma^2)$$
$$R_t(s,a) = \bar{R}_t(s,a) + v_t \qquad\qquad v_t \sim \mathcal{N}(0,\varepsilon^2)$$

Each transition had (at most) three possible outcome, with probabilities 0.6, 0.3, and 0.1, assigned randomly with replacement from the 8 states. In order to avoid bias related to the exploration policy, each learning algorithm observed the same set of 1000 choices (chosen according to the objectively optimal policy, plus softmax decision noise), and the greedy policy resulting from its learned values was assessed according to the true $\bar{R}$ values at that point. The entire process was repeated 5000 times for each different setting of $\sigma$ and $\varepsilon$ parameters.

We compared the performance of a model-based approach using value iteration with a fixed, optimal reward learning rate and transition counting (MB) against various model-free algorithms including Q(0), SARSA(0), and SARSA(1) (with fixed optimal learning rates), all using a discount factor of $\gamma = 0.9$. As expected, all learners showed a decrement in reward as $\sigma$ increased. Figure 2 shows the difference in mean reward obtained between MB and SARSA(0). Q(0) and SARSA(1) showed the same pattern of results.

The correspondence between the theoretical results and the simulation confirms that the theoretical findings do hold more generally, and we claim that the same underlying effects drive these results.

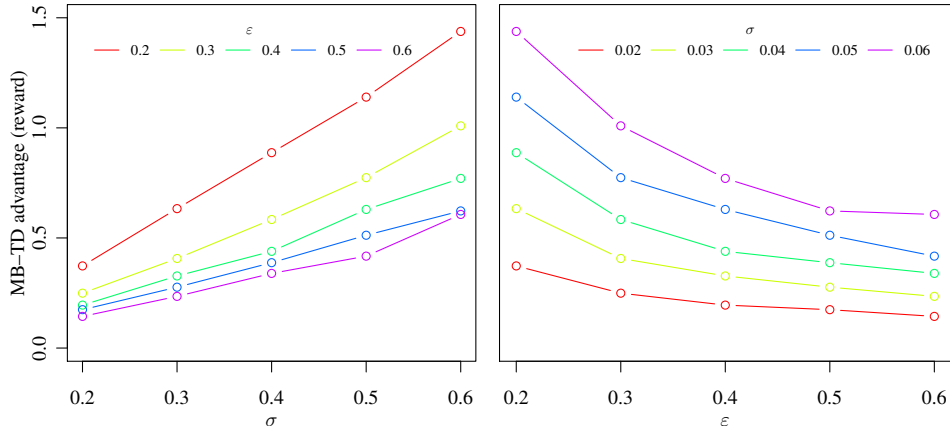

Figure 2: Difference in reward obtained between MB and SARSA(0)

# 4 Human behavior

Human subjects performed a decision task that represented an MDP with 4 states and 2 actions. The rewards followed the same contractive Gaussian diffusion process used in section 3, with $\sigma$ and $\varepsilon$ parameters varied across subjects. We sought changes in the reliance on model-based and model-free strategies via regressions of past events onto current choices [21]. We hypothesized that model-based RL would be uniquely favored for large $\sigma$ and small $\varepsilon$.

## 4.1 Methods

### 4.1.1 Participants

55 individuals from the undergraduate subject pool and the surrounding community participated in the experiment. Twelve received monetary compensation based on performance, and the remainder received credit fulfilling course requirements. All participants gave informed consent and the study was approved by the human subjects ethics board of the institute.

### 4.1.2 Task

Subjects viewed a graphical representation of a rotating disc with four pairs of colored squares equally spaced around the edge. Each pair of squares constituted a state ($s \in \mathcal{S} = \{\mathsf{N}, \mathsf{E}, \mathsf{S}, \mathsf{W}\}$) and had a unique distinguishable color and icon indicating direction (an arrow of some type). Each of the two squares in a state represented an action ($a \in \mathcal{A} = \{\mathsf{L}, \mathsf{R}\}$), and had a left- or right-directed icon. During the task, only the top quadrant of the disc was visible at any time, and at decision time subjects could select the left or right action by pressing the left or right arrow button on a keyboard. Immediately after selecting an action, between zero and five coins (including a pie-fraction of a coin) appeared under the selected action square, representing a reward ($R \in [0, 5]$). After 600 ms, the disc began rotating and the reward became slowly obscured over the next 1150 ms until a new pair of squares was at the top of the disc and the next decision could be entered, as seen in Figure 3.

The state dynamics were determined by a fixed transition function ($T : \mathcal{S} \times \mathcal{A} \to \mathcal{A}$) such that each action was most likely to lead to the next adjacent state along the edge of the disc (e.g., $T(\mathsf{N}, \mathsf{L}) = \mathsf{W}$). To this, additional uniform outcome noise was added with probability $0.4$. The reward distribution followed the same Gaussian process given in the previous sections, except shifted and trimmed. The parameters $\sigma$ and $\varepsilon$ were varied by condition.

$$T : \mathcal{S} \times \mathcal{A} \times \mathcal{S} \to [0, 1] \qquad T(s, a, s') = \begin{cases} 0.7 & \text{if } s' = T(s, a) \\ 0.1 & \text{otherwise} \end{cases}$$

$$R_t : \mathcal{S} \times \mathcal{A} \to [0, 5] \qquad R_t(s, a) = \min(\max(\bar{R}_t(s, a) + v_t + 2.5, 0), 5)$$

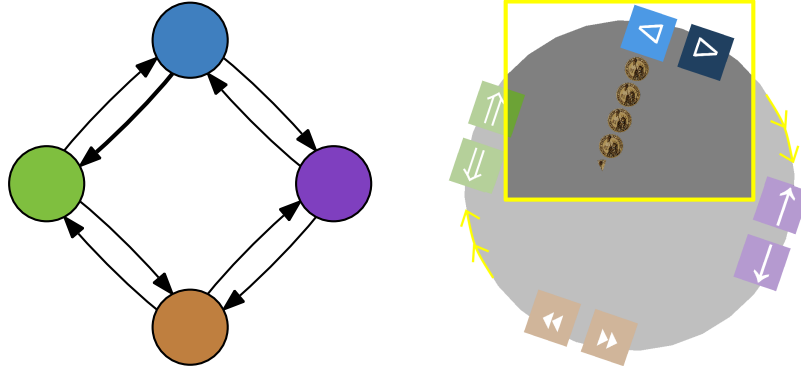

Figure 3: Abstract task layout and screen shot shortly after a choice is made (yellow box indicates visible display): Each state has two actions, right (red) and left (blue), which lead to the indicated state with 70% probability, and otherwise to another state at random. Each action also results in a reward of 0–5 coins.

Each subject was first trained on the transition and reward dynamics of the task, including 16 observations of reward samples where the latent value $\bar{R}$ was shown so as to get a feeling for both the change and noise processes. They then performed 500 choice trials in a single condition. Each subject was randomly assigned to one of 12 conditions, made up of $\sigma \in \{0.03, 0.0462, 0.0635, 0.0882, 0.1225, 0.1452\}$ partially crossed with $\varepsilon \in \{0, 0.126, 0.158, 0.316, 0.474, 0.506\}$.

### 4.1.3 Analysis

Because they use different sampling strategies to estimate action values, TD and model-based RL differ in their predictions of how experience with states and rewards should affect subsequent choices. Here, we use a regression analysis to measure the extent to which choices at a state are influenced by recent previous events characteristic of either approach [21]. This approach has the advantage of making only very coarse assumptions about the learning process, as opposed to likelihood-based model-fits which may be biased by the specific learning equations assumed. By confining our analyses to the most recent samples we remain agnostic about free parameters with non-linear effects such as learning rates and discount factors, but rather measure the relative strength of reliance on either sort of evidence directly using a general linear model. Regardless of the actual learning process, the most recent sample should have the strongest effect [22]. Accordingly, below we define explanatory variables that capture the most recently experienced reward sample that would be relevant to a choice under either Q(1) TD or model-based planning.

The data for each subject were considered to be the sequence of states visited, $S_t$, actions taken, $A_t$, and rewards received, $R_t$. We define additional vector time sequences $a$, $j$, $r$, $q$, and $p$, each indexed by time and state and referred to generally as $x_t(s)$, with all $x_0$ initially undefined. For each observation we perform the following updates:

$$
\begin{aligned}
w_t &= [A_t = a_t(S_t)] & \text{`stay' vs. `switch' (boolean indicator)} \\
a_{t+1}(S_t) &= A_t & \text{last action} \\
j_{t+1}(S_t) &= [S_{t+1} \neq T(S_t, A_t)] & \text{`jump' unexpected transition} \\
r_{t+1}(S_t) &= R_t & \text{immediate reward} \\
q_{t+1}(S_{t-1}) &= R_t & \text{subsequent reward} \\
p_{t+1}(S_t) &= r_{t+1}(T(S_t, A_t)) & \text{expected reward} \\
x_{t+1}(s) &= x_t(s) \,\forall s \neq S_t & \text{for } x = a, j, r, q, \text{ and } p \\
d_{t+1} &= |R_t - r_t| & \text{change}
\end{aligned}
$$

For convenience, we use $x_t$ to mean $x_t(S_t)$. Note that these vectors are step functions, such that each value is updated ($x_t \neq x_{t-1}$) only when a relevant observation is made. They thus always represent the most recent relevant sample.

Given the task dynamics, we can consider how a TD-based Q-learning system and a model-based planning system would compute values. Both take into account the last sample of the immediate reward, $r_t$. They differ in how they account for the reward from the "next state": either, for Q(1), as $q_t$ (the last reward received from the state visited after the last visit to $S_t$) or, for model-based RL, as $p_t$ (the last sample of the reward at the true successor state). That is, while TD(1) will incorporate the reward observed following $R_t$, regardless of the state, a model-based system will instead consider the expected successor state [21]. While the latter two reward observations will be the same in some cases, they can disagree either after a jump trial ($j$, where the model-based and sample successor states differ), or when the successor state has more recently been visited from a different predecessor state (providing a reward sample known to model-based but not to TD).

Given this, we can separate the effects of model-based and model-free learning by defining additional explanatory variables:

$$r'_t = \begin{cases} q_t & \text{if } q_t = p_t \\ 0 & \text{otherwise (after mean correction)} \end{cases} \qquad \text{common}$$

$$q^*_t = q_t - r'_t \qquad \text{unique}$$

$$p^*_t = p_t - r'_t$$

While $r'$ represents the cases where the two systems use the same reward observation, $q^*$ and $p^*$ are the residual rewards unique to each learning system.

We applied a mixed-effects logistic regression model using `glmer` [23] to predict 'stay' ($w_t = 0$) trials. Any regressors of interest were mean-corrected before being entered into the design. Any trial in which one of the variables was undefined (e.g., the first visit to a state) was omitted. Also, we required that subjects have at least 50 (10%) switch trials to be included.

First we examined the main effects with a regression including fixed effects of interest for $r$, $r'$, $q^*$, $p^*$, and random effects of no interest for $r$, $q$, and $p$ (without covariances). Next, we ran a regression adding all the interactions between the condition variables ($\sigma$, $\varepsilon$) and the specific reward effects ($q^*$, $p^*$). Finally, we additionally included the interaction between change in reward on the previous trial ($d$) and the specific reward effects.

## 4.2 Results

A total of 5 subjects failed to meet the inclusion criterion of 50 switch trials (in each case because they pressed the same button on almost all trials), leaving 500 decision trials from each of 50 subjects. Subjects were observed to switch on $143 \pm 55$ trials (mean $\pm$ 1 SD). As designed, there were an average of $151 \pm 17$ 'jump' trials per subject. The number of trials in which TD and model-based disagreed as to the most recent relevant sample of the next-state reward ($r' = 0$) was $243 \pm 26$, and for $181 \pm 19$ of these, it was due to a more recent visit to the next state. The results of the regressions are shown in Table 1.

Beyond the trivial effects of perseveration and reward, subjects showed a substantial amount of TD-type learning ($q^* > 0$), and a smaller but significant amount of model-based lookahead ($p^* > 0$). The interactions of these effects by condition demonstrated that subjects in higher drift conditions showed significantly less TD ($\sigma \times q^* < 0$) but unreduced model-based learning ($\sigma \times p^*$), possibly due to the relative disadvantage of TD with increased drift. Similarly, higher noise conditions showed decreased model-based effects ($\varepsilon \times p^* < 0$) and no change in TD, which may be driven by the decreasing advantage of MB. Note that, since the (nonsignificant) trend on the unaffected variable is positive, it is unlikely that either interaction effect results from a nonspecific change in performance or the "noisiness" of choices. Both of these effects are consistent with the pattern of differential reliance predicted by the theoretical analysis. The effect of change on the previous trial ($d$) provides one hint as to how subjects may adjust their reliance on either system dynamically: higher changes are indicative of noisier environments which are thus expected to promote TD learning.

## 5 Discussion

We have shown that humans systematically adjust their reliance on learning approaches according to the statistics of the task, in a way qualitatively consistent with the theoretical considerations

Table 1: Behavioral effects from nested regressions (each including preceding groups)

| variable | effects | $z$ | $p$ | description |
|---|---|---|---|---|
| constant | mixed | 11.61 ⇑ | $10^{-29}$ | perseveration |
| $r$ | mixed | 14.99 ⇑ | $10^{-49}$ | last immediate r |
| $r'$ | mixed | 5.55 ⇑ | $10^{-7}$ | common next r |
| $q^*$ | mixed | 6.40 ⇑ | $10^{-9}$ | TD(1) next-step r |
| $p^*$ | mixed | 2.51 ↑ | 0.012 | model predicted r |
| $\sigma \times q^*$ | fixed | -4.07 ⇓ | 0.00005 | TD with change |
| $\sigma \times p^*$ | fixed | 0.67 | 0.50 | model with change |
| $\varepsilon \times q^*$ | fixed | 0.99 | 0.32 | TD with noise |
| $\varepsilon \times p^*$ | fixed | -2.11 ↓ | 0.035 | model with noise |
| $d \times q^*$ | mixed | 1.63 | 0.10 | TD after change |
| $d \times p^*$ | mixed | -3.06 ↓ | 0.0022 | model after change |

presented. Model-based methods, while always superior to TD in terms of performance, have the largest advantage in the presence of change paired with low environmental noise, because the Monte Carlo sampling strategy of TD interferes with tracking fast change. If the additional costs of model-based computation are fixed, this would motivate employing the system only in the regime where its advantage was most pronounced [18]. Consistent with this, human behavior exhibited relatively larger use of model-based RL with increased reward volatility and lesser use of it with increased observation noise.

Of course, increasing either the volatility or noise parameters makes the task harder, and a decline in the marker for either sort of learning, as we observed, implies an overall decrement in performance. However, as the decrement was specific to one or the other explanatory variable, this may also be interpreted as a relative increase in use of the unaffected strategy. It is also worth noting that the linearized regression analysis examines only the effect of the most recent rewards, and the weighting of those relative to earlier samples will depend on the learning rate [22]. Thus a decrease in learning rate for either system may be confounded with a decrease in the strength of its effect in our analysis. However, while the optimal learning rates are also predicted to differ between conditions, these predictions are common to both systems, and it seems unlikely that each would differentially adjust its learning rate in response to a different manipulation.

The characteristics associated with these learning systems in psychology can be seen as consequences of the relative strengths of model-based and model-free learning. If the model-based system is most useful in conditions of low noise and high volatility, then the appropriate learning rates for such a system are large: there is less need and utility to take multiple samples for the purpose of averaging. In this case of a high learning rate, model-based learning is closely aligned with single-shot episodic encoding, possibly subsuming such a system [17], as well as with learning categorical, verbalizable rules in the psychological sense, rather than averages. This may also explain the selective engagement of putatively model-based brain regions such as the dorsolateral prefrontal cortex in tasks with less stochastic outcomes [24]. Finally, this relates indirectly to the well known phenomenon whereby dominance shifts from the model-based to the model-free controller with overtraining: a model-based system dominates early not simply because it learns faster, but because it is capable of better learning with fewer trials.

The specific advantage of high learning rates may well motivate the brain to use a restricted model-based system, such as one with learning rate fixed to 1. Indeed (see Supplemental materials), this restriction has little detriment on the system's advantage over TD in the circumstances where it would be expected to be used, but causes drastic performance problems as observation noise increases, since averaging over samples is then required. Such a limitation might have useful computational advantages. Transition matrices learned this way, for instance, will be sparse: just records of trajectories. Such matrices admit both compressed representations and more efficient planning algorithms (e.g., tree search) as, in the fully deterministic case, only one trajectory must be examined. Conversely, evaluations in a model based system are extremely costly when transitions are highly stochastic, since averages must be computed over exponentially many paths, while they add no cost to model-free learning.

**Acknowledgments** This work was supported by Award Number R01MH087882 from NIMH as part of the NSF/NIH CRCNS Program, and by a Scholar Award from the McKnight Foundation.

## References

[1] Bernard W. Balleine, Nathaniel D. Daw, and John P. O'Doherty. Multiple forms of value learning and the function of dopamine. In Paul W. Glimcher, Colin F. Camerer, Ernst Fehr, and Russell A. Poldrack, editors, *Neuroeconomics: Decision Making and the Brain*, chapter 24, pages 367–387. Academic Press, London, 2008.

[2] Antoine Bechara. Decision making, impulse control and loss of willpower to resist drugs: a neurocognitive perspective. *Nat Neurosci*, 8(11):1458–63, 2005.

[3] Frederick Toates. The interaction of cognitive and stimulus-response processes in the control of behaviour. *Neuroscience & Biobehavioral Reviews*, 22(1):59–83, 1997.

[4] Peter Dayan. Goal-directed control and its antipodes. *Neural Netw*, 22:213–219, 2009.

[5] Neal Schmitt, Bryan W. Coyle, and Larry King. Feedback and task predictability as determinants of performance in multiple cue probability learning tasks. *Organ Behav Hum Perform*, 16(2):388–402, 1976.

[6] Berndt Brehmer and Jan Kuylenstierna. Task information and performance in probabilistic inference tasks. *Organ Behav Hum Perform*, 22:445–464, 1978.

[7] B J Knowlton, L R Squire, and M A Gluck. Probabilistic classification learning in amnesia. *Learn Mem*, 1(2):106–120, 1994.

[8] W. Todd Maddox and F. Gregory Ashby. Dissociating explicit and procedural-learning based systems of perceptual category learning. *Behavioural Processes*, 66(3):309–332, 2004.

[9] W. Todd Maddox, J. Vincent Filoteo, Kelli D. Hejl, and A. David Ing. Category number impacts rule-based but not information-integration category learning: Further evidence for dissociable category-learning systems. *J Exp Psychol Learn Mem Cogn*, 30(1):227–245, 2004.

[10] R. A. Poldrack, J. Clark, E. J. Paré-Blagoev, D. Shohamy, J. Creso Moyano, C. Myers, and M. A. Gluck. Interactive memory systems in the human brain. *Nature*, 414(6863):546–550, 2001.

[11] Bernard W. Balleine and Anthony Dickinson. Goal-directed instrumental action: contingency and incentive learning and their cortical substrates. *Neuropharmacology*, 37(4–5):407–419, 1998.

[12] Kenji Doya. What are the computations of the cerebellum, the basal ganglia and the cerebral cortex? *Neural Netw*, 12(7–8):961–974, 1999.

[13] Nathaniel D. Daw, Yael Niv, and Peter Dayan. Uncertainty-based competition between prefrontal and dorsolateral striatal systems for behavioral control. *Nat Neurosci*, 8(12):1704–1711, 2005.

[14] Ben Seymour, John P. O'Doherty, Peter Dayan, Martin Koltzenburg, Anthony K. Jones, Raymond J. Dolan, Karl J. Friston, and Richard S. Frackowiak. Temporal difference models describe higher-order learning in humans. *Nature*, 429(6992):664–667, 2004.

[15] John P. O'Doherty, Peter Dayan, Johannes Schultz, Ralf Deichmann, Karl Friston, and Raymond J. Dolan. Dissociable roles of ventral and dorsal striatum in instrumental conditioning. *Science*, 304(5669):452–454, 2004.

[16] Adam Johnson and A. David Redish. Hippocampal replay contributes to within session learning in a temporal difference reinforcement learning model. *Neural Netw*, 18(9):1163–1171, 2005.

[17] Máté Lengyel and Peter Dayan. Hippocampal contributions to control: The third way. In J.C. Platt, D. Koller, Y. Singer, and S. Roweis, editors, *Advances in Neural Information Processing Systems 20*, pages 889–896. MIT Press, Cambridge, MA, 2008.

[18] Mehdi Keramati, Amir Dezfouli, and Payam Piray. Speed/accuracy trade-off between the habitual and the goal-directed processes. *PLoS Comput Biol*, 7(5):e1002055, 2011.

[19] Michael Kearns and Satinder Singh. Finite-sample convergence rates for q-learning and indirect algorithms. In Michael S. Kearns, Sara A. Solla, and David A. Cohn, editors, *Advances in Neural Information Processing Systems 11*, volume 11, pages 996–1002. MIT Press, Cambridge, MA, 1999.

[20] R. E. Kalman. A new approach to linear filtering and prediction problems. *J Basic Eng*, 82(1):35–45, 1960.

[21] Nathaniel D Daw, S. J. Gershman, B. Seymour, P. Dayan, and R. J. Dolan. Model-based influences on humans' choices and striatal prediction errors. *Neuron*, 69(6):1204–1215, 2011.

[22] Brian Lau and Paul W Glimcher. Dynamic response-by-response models of matching behavior in rhesus monkeys. *J Exp Anal Behav*, 84(3):555–579, 2005.

[23] Douglas Bates, Martin Maechler, and Ben Bolker. *lme4: Linear mixed-effects models using S4 classes*, 2011. R package version 0.999375-39.

[24] Saori C Tanaka, Kazuyuki Samejima, Go Okada, Kazutaka Ueda, Yasumasa Okamoto, Shigeto Yamawaki, and Kenji Doya. Brain mechanism of reward prediction under predictable and unpredictable environmental dynamics. *Neural Netw*, 19(8):1233–1241, 2006.

